# Phoneme Classification using Constrained Variational Gaussian Process Dynamical System

**Hyunsin Park**
Department of EE, KAIST
Daejeon, South Korea
hs.park@kaist.ac.kr

**Sungrack Yun**
Qualcomm Korea
Seoul, South Korea
sungrack@qualcomm.com

**Sanghyuk Park**
Department of EE, KAIST
Daejeon, South Korea
shine0624@kaist.ac.kr

**Jongmin Kim**
Department of EE, KAIST
Daejeon, South Korea
kimjm0309@gmail.com

**Chang D. Yoo**
Department of EE, KAIST
Daejeon, South Korea
cdyoo@ee.kaist.ac.kr

## Abstract

For phoneme classification, this paper describes an acoustic model based on the variational Gaussian process dynamical system (VGPDS). The nonlinear and nonparametric acoustic model is adopted to overcome the limitations of classical hidden Markov models (HMMs) in modeling speech. The Gaussian process prior on the dynamics and emission functions respectively enable the complex dynamic structure and long-range dependency of speech to be better represented than that by an HMM. In addition, a variance constraint in the VGPDS is introduced to eliminate the sparse approximation error in the kernel matrix. The effectiveness of the proposed model is demonstrated with three experimental results, including parameter estimation and classification performance, on the synthetic and benchmark datasets.

## 1   Introduction

Automatic speech recognition (ASR), the process of automatically translating spoken words into text, has been an important research topic for several decades owing to its wide array of potential applications in the area of human-computer interaction (HCI). The state-of-the-art ASR systems typically use hidden Markov models (HMMs) [1] to model the sequential articulator structure of speech signals. There are various issues to consider in designing a successful ASR and certainly the following two limitations of an HMM need to be overcome. 1) An HMM with a first-order Markovian structure is suitable for capturing short-range dependency in observations and speech requires a more flexible model that can capture long-range dependency in speech. 2) Discrete latent state variables and sudden state transitions in an HMM have limited capacity when used to represent the continuous and complex dynamic structure of speech. These limitations must be considered when seeking to improve the performance of an ASR.

To overcome these limitations, various models have been considered to model the complex structure of speech. For example, the stochastic segment model [2] is a well-known generalization of the HMM that represents long-range dependency over observations using a time-dependent emission function. And the hidden dynamical model [3] is used for modeling the complex nonlinear dynamics of a physiological articulator.

Another promising research direction is to consider a nonparametric Bayesian model for nonlinear probabilistic modeling of speech. Owing to the fact that nonparametric models do not assume any

fixed model structure, they are generally more flexible than parametric models and can allow dependency among observations naturally. The Gaussian process (GP) [4], a stochastic process over a real-valued function, has been a key ingredient in solving such problems as nonlinear regression and classification. As a standard supervised learning task using the GP, Gaussian process regression (GPR) offers a nonparametric Bayesian framework to infer the nonlinear latent function relating the input and the output data. Recently, researchers have begun focusing on applying the GP to unsupervised learning tasks with high-dimensional data, such as the Gaussian process latent variable model (GP-LVM) for reduction of dimensionality [5-6]. In [7], a variational inference framework was proposed for training the GP-LVM. The variational approach is one of the sparse approximation approaches [8]. The framework was extended to the variational Gaussian process dynamical system (VGPDS) in [9] by augmenting latent dynamics for modeling high-dimensional time series data. High-dimensional time series have been incorporated in many applications of machine learning such as robotics (sensor data), computational biology (gene expression data), computer vision (video sequences), and graphics (motion capture data). However, no previous work has considered the GP-based approach for speech recognition tasks that involve high-dimensional time series data.

In this paper, we propose a GP-based acoustic model for phoneme classification. The proposed model is based on the assumption that the continuous dynamics and nonlinearity of the VGPDS can be better represent the statistical characteristic of real speech than an HMM. The GP prior over the emission function allows the model to represent long-range dependency over the observations of speech, while the HMM does not. Furthermore, the GP prior over the dynamics function enables the model to capture the nonlinear dynamics of a physiological articulator.

Our contributions are as follows: 1) we introduce a GP-based model for phoneme classification tasks for the first time, showing that the model has the potential of describing the underlying characteristics of speech in a nonparametric way; 2) we propose a prior for hyperparameters and a variance constraint that are specially designed for ASR; and 3) we provide extensive experimental results and analyses to reveal clearly the strength of our proposed model.

The remainder of the paper is structured as follows: Section 2 introduces the proposed model after a brief description of the VGPDS. Section 3 provides extensive experimental evaluations to prove the effectiveness of our model, and Section 4 concludes the paper with a discussion and plans for future work.

## 2 Acoustic modeling using Gaussian Processes

### 2.1 Variational Gaussian Process Dynamical System

The VGPDS [9] models time series data by assuming that there exist latent states that govern the data. Let $\mathbf{Y} = [[y_{11}, \cdots y_{N1}]^T, \cdots, [y_{1D}, \cdots y_{ND}]^T] \in \mathbb{R}^{N \times D}$, $\mathbf{t} = [t_1, \cdots, t_N]^T \in \mathbb{R}_+^N$, and $\mathbf{X} = [[x_{11}, \cdots x_{N1}]^T, \cdots, [x_{1Q}, \cdots x_{NQ}]^T] \in \mathbb{R}^{N \times Q}$ be observed data, time, and corresponding latent state, where $N$, $D$, and $Q(< D)$ are the number of samples, the dimension of the observation space, and the dimension of the latent space, respectively. In the VGPDS, these variables are related as follows:

$$
\begin{aligned}
x_{nj} &= g_j(t_n) + \eta_{nj}, \quad \eta_{nj} \sim \mathcal{N}(0, 1/\beta_j^x), \\
y_{ni} &= f_i(\mathbf{x}_n) + \epsilon_{ni}, \quad \epsilon_{ni} \sim \mathcal{N}(0, 1/\beta_i^y),
\end{aligned}
\tag{1}
$$

where $f_i(\mathbf{x}) \sim \mathcal{GP}(\mu_i^f(\mathbf{x}), k_i^f(\mathbf{x}, \mathbf{x}'))$ and $g_j(t) \sim \mathcal{GP}(\mu_j^g(t), k_j^g(t, t'))$ are the emission function from the latent space to the $i$-th dimension of the observation space and the dynamics function from the time space to the $j$-th dimension of the latent space, respectively. Here, $n \in \{1, \cdots, N\}$, $i \in \{1, \cdots, D\}$, and $j \in \{1, \cdots, Q\}$. In this paper, a zero-mean function is used for all GPs. Fig. 1 shows graphical representations of HMM and VGPDS. Although the Gaussian process dynamical model (GPDM) [10], which involves an auto-regressive dynamics function, is also a GP-based model for time-series, it is not considered in this paper.

The marginal likelihood of the VGPDS is given as

$$
p(\mathbf{Y}|\mathbf{t}) = \int p(\mathbf{Y}|\mathbf{X})p(\mathbf{X}|\mathbf{t})d\mathbf{X}.
\tag{2}
$$

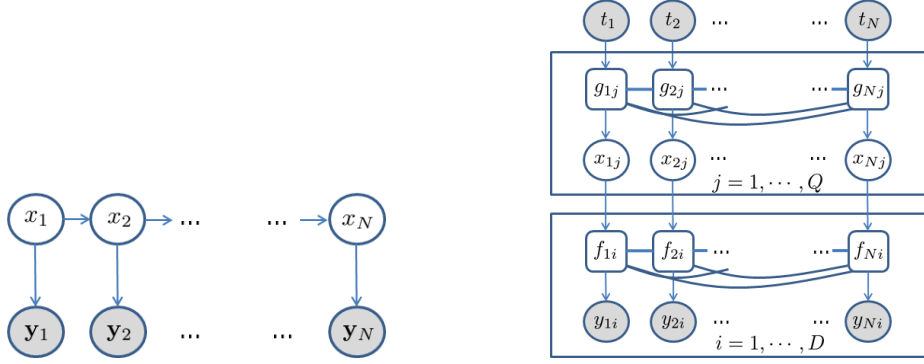

Figure 1: Graphical representations of (left) the left-to-right HMM and (right) the VGPDS: In the left figure, $\mathbf{y}_n \in \mathbb{R}^D$ and $x_n \in \{1, \cdots, C\}$ are observations and discrete latent states. In the right figure, $y_{ni}$, $f_{ni}$, $x_{nj}$, $g_{nj}$, and $t_n$ are observations, emission function points, latent states, dynamics function points, and times, respectively. All function points in the same plate are fully connected.

Since the integral in Eq. (2) is not tractable, a variational method is used by introducing a variational distribution $q(\mathbf{X})$. A variational lower bound on the logarithm of the marginal likelihood is

$$
\begin{aligned}
\log p(\mathbf{Y}|\mathbf{t}) &\geq \int q(\mathbf{X}) \log \frac{p(\mathbf{Y}|\mathbf{X})p(\mathbf{X}|\mathbf{t})}{q(\mathbf{X})} d\mathbf{X} \\
&= \int q(\mathbf{X}) \log p(\mathbf{Y}|\mathbf{X}) d\mathbf{X} - \int q(\mathbf{X}) \log \frac{q(\mathbf{X})}{p(\mathbf{X}|\mathbf{t})} d\mathbf{X} \\
&= \mathcal{L} - \mathrm{KL}(q(\mathbf{X})||p(\mathbf{X}|\mathbf{t})).
\end{aligned}
\tag{3}
$$

By the assumption of independence over the observation dimension, the first term in Eq. (3) is given as

$$
\mathcal{L} = \sum_{i=1}^{D} \int q(\mathbf{X}) \log p(\mathbf{y}_i|\mathbf{X}) d\mathbf{X} = \sum_{i=1}^{D} \mathcal{L}_i.
\tag{4}
$$

In [9], a variational approach which involves sparse approximation of the covariance matrix obtained from GP is proposed. The variational lower bound on $\mathcal{L}_i$ is given as

$$
\mathcal{L}_i \geq \log \left[ \frac{(\beta_i^y)^{N/2} |\tilde{\mathbf{K}}_i|^{1/2}}{(2\pi)^{N/2} |\beta_i^y \Psi_{2i} + \tilde{\mathbf{K}}_i|^{1/2}} e^{(-\frac{1}{2} \mathbf{y}_i^T \mathbf{W}_i \mathbf{y}_i)} \right] - \frac{\beta_i^y}{2} (\psi_{0i} - \mathrm{Tr}(\tilde{\mathbf{K}}_i^{-1} \Psi_{2i})),
\tag{5}
$$

where $\mathbf{W}_i = \beta_i^y \mathbf{I}_N - (\beta_i^y)^2 \Psi_{1i} (\beta_i^y \Psi_{2i} + \tilde{\mathbf{K}}_i)^{-1} \Psi_{1i}^T$. Here, $\tilde{\mathbf{K}}_i \in \mathbb{R}^{M \times M}$ is a kernel matrix calculated using the $i$-th kernel function and inducing input variables $\tilde{\mathbf{X}} \in \mathbb{R}^{M \times Q}$ that are used for sparse approximation of the full kernel matrix $\mathbf{K}_i$. The closed-form of the statistics $\{\psi_{0i}, \Psi_{1i}, \Psi_{2i}\}_{i=1}^{D}$, which are functions of variational parameters and inducing points, can be found in [9]. In the second term of Eq. (3), $p(\mathbf{X}|\mathbf{t}) = \prod_{j=1}^{Q} p(\mathbf{x}_j)$ and $q(\mathbf{X}) = \prod_{n}^{N} \prod_{j=1}^{Q} \mathcal{N}(\mu_{nj}, s_{nj})$ are the prior for the latent state and the variational distribution that is used for approximating the posterior of the latent state, respectively.

The parameter set $\mathbf{\Theta}$, which consists of the hyperparameters $\{\boldsymbol{\theta}^f, \boldsymbol{\theta}^g\}$ of the kernel functions, the noise variances $\{\boldsymbol{\beta}^y, \boldsymbol{\beta}^x\}$, the variational parameters $\{[\mu_{n1}, \cdots, \mu_{nQ}], [s_{n1}, \cdots, s_{nQ}]\}_{n=1}^{N}$ of $q(\mathbf{X})$, and the inducing input points $\tilde{\mathbf{X}}$, is estimated by maximizing the lower bound on $\log p(\mathbf{Y}|\mathbf{t})$ in Eq. (3) using a scaled conjugate gradient (SCG) algorithm.

## 2.2 Acoustic modeling using VGPDS

For several decades, HMM has been the predominant model for acoustic speech modeling. However, as we mentioned in Section 1, the model suffers from two major limitations: discrete state variables and first-order Markovian structure which can model short-range dependency over the observations.

To overcome such limitations of the HMM, we propose an acoustic speech model based on the VGPDS, which is a nonlinear and nonparametric model that can be used to represent the complex dynamic structure of speech and long-range dependency over observations of speech. In addition, to fit the model to large-scale speech data, we describe various implementation issues.

### 2.2.1 Time scale modification

The time length of each phoneme segment in an utterance varies with various conditions such as position of the phoneme segment in the utterance, emotion, gender, and other speaker and environment conditions. To incorporate this fact into the proposed acoustic model, the time points $t_n$ are modified as follows:

$$t_n = \frac{n-1}{N-1}, \tag{6}$$

where $n$ and $N$ are the observation index and the number of observations in a phoneme segment, respectively. This time scale modification makes all phoneme signals have unit time length.

### 2.2.2 Hyperparameters

To compute the kernel matrices in Eq. (5), the kernel function must be defined. We use the radial basis function (RBF) kernel for the emission function $f$ as follows:

$$k^f(\mathbf{x}, \mathbf{x}') = \alpha^f \exp\left(-\sum_{j=1}^{Q} \omega_j^f (x_j - x_j')^2\right), \tag{7}$$

where $\alpha^f$ and $\omega_j^f$ are the RBF kernel variance and the $j$-th inverse length scale, respectively. The RBF kernel function is adopted for representing smoothness of speech. For the dynamics function $g$, the following kernel function is used:

$$k^g(t, t') = \alpha^g \exp\left(-\omega^g(t - t')^2\right) + \lambda t t' + b, \tag{8}$$

where $\lambda$ and $b$ are linear kernel variance and bias, respectively. The above dynamics kernel, which consists of both linear and nonlinear components, is used for representing the complex dynamics of the articulator. All hyperparameters are assumed to be independent in this paper.

In [11], same kernel function parameters are shared over all dimensions of human-motion capture data and high-dimensional raw video data. However, this extensive sharing of the hyperparameters is unsuitable for speech modeling. Even though each dimension of observations is normalized in advance to have unit variance, the signal-to-noise ratio (SNR) is not consistent over all dimensions. To handle this problem, this paper considers each dimension to be modeled independently using different kernel function parameters. Therefore, the hyperparameter sets are defined as $\boldsymbol{\theta}^f = \{\alpha_i^f, \{\omega_{1i}^f, \cdots, \omega_{Qi}^f\}\}_{i=1}^{D}$ and $\boldsymbol{\theta}^g = \{\alpha_j^g, \omega_j^g, \lambda_j, b_j\}_{j=1}^{Q}$.

### 2.2.3 Priors on the hyperparameters

In the parameter estimation of the VGPDS, the SCG algorithm does not guarantee the optimal solution. To overcome this problem, we place the following prior on the hyperparameters of the kernel functions as given below

$$p(\gamma) \propto \exp(-\gamma^2/\bar{\gamma}), \tag{9}$$

where $\gamma \in \{\boldsymbol{\theta}^f, \boldsymbol{\theta}^g\}$ and $\bar{\gamma}$ are the hyper-parameter and the model parameter of the prior, respectively. In this paper, $\bar{\gamma}$ is set to the sample variance for the hyperparameters of the emission kernel functions, and $\bar{\gamma}$ is set to 1 for the hyperparameters of the dynamics kernel functions. Uniform priors are adopted for other hyperparameters, then the parameters of the VGPDS are estimated by maximizing the joint distribution $p(\mathbf{Y}, \boldsymbol{\Theta}|\mathbf{t}) = p(\mathbf{Y}|\mathbf{t}, \boldsymbol{\Theta})p(\boldsymbol{\Theta})$.

### 2.2.4 Variance constraint

In the lower bound of Eq. (5), the second term on the right-hand side is the regularization term that represents the sparse approximation error of the full kernel matrix $\mathbf{K}_i$. Note that with more inducing

input points, approximation error becomes smaller. However, only a small number of inducing input points can be used owing to the limited availability of computational power, which increases the effect of the regularization term.

To mitigate this problem, we introduce the following constraint on the diagonal terms of the covariance matrix as given below:

$$\frac{\text{Tr}(\langle \mathbf{K}_i \rangle_{q(\mathbf{X})})}{N} + 1/\beta_i^y = \sigma_i^2, \tag{10}$$

where $\langle \mathbf{K}_i \rangle_{q(\mathbf{X})}$ and $\sigma_i^2$ are the expectation of the full kernel matrix $\mathbf{K}_i$ and the sample variance of the $i$-th dimension of the observation, respectively. This constraint is designed so that the variance of each observation calculated from the estimated model is equal to the sample variance. By using $\psi_{0i} = \text{Tr}(\langle \mathbf{K}_i \rangle_{q(\mathbf{X})})$, the inverse noise variance parameter is obtained directly by $\beta_i^y = (\sigma_i^2 - \psi_{0i}/N)^{-1}$ without separate gradient-based optimization. Then, the partial derivative $\frac{\partial \log \beta_i^y}{\partial \psi_{0i}} = \frac{1}{N\sigma^2 - \psi_{0i}}$ is used for SCG-based optimization. In Section 3.1, the effectiveness of the variance constraint is demonstrated empirically.

### 2.3 Classification

For classification with trained VGPDSs, maximum-likelihood (ML) decoding is used. Let $\mathcal{D}^{(l)} = \{\mathbf{Y}^{(l)}, \mathbf{t}^{(l)}\}$ and $\Theta^{(l)}$ be the observation and parameter sets of the $l$-th VGPDS, respectively. Given the test data $\mathcal{D}_* = \{\mathbf{Y}_*, \mathbf{t}_*\}$, the classification result $\hat{l} \in \{1, \cdots, L\}$ can be obtained by

$$
\begin{aligned}
\hat{l} &= \arg\max_l \log p(\mathbf{Y}_*|\mathbf{t}_*, \mathbf{Y}^{(l)}, \mathbf{t}^{(l)}, \Theta^{(l)}) \\
&= \arg\max_l \log \frac{p(\mathbf{Y}^{(l)}, \mathbf{Y}_*|\mathbf{t}^{(l)}, \mathbf{t}_*, \Theta^{(l)})}{p(\mathbf{Y}^{(l)}|\mathbf{t}^{(l)}, \Theta^{(l)})}.
\end{aligned} \tag{11}
$$

## 3 Experiments

To evaluate the effectiveness of the proposed model, three different kinds of experiments have been designed:

1. *Parameter estimation*: validating the effectiveness of the proposed variance constraint (Section 2.2.4) on model parameter estimation
2. *Two-class classification using synthetic data*: demonstrating explicitly the advantages of the proposed model over the HMM with respect to the degree of dependency over the observations
3. *Phoneme classification*: evaluating the performance of the proposed model on real speech data

Each experiment is described in detail in the following subsections. In this paper, the proposed model is referred to as the constrained-VGPDS (CVGPDS).

### 3.1 Parameter estimation

In this subsection, the experiments of parameter estimation on synthetic data are described. Synthetic data are generated by using a phoneme model that is selected from the trained models in Section 3.3 and then modified. The RBF kernel variances of the emission functions and the emission noise variances are modified from the selected model. In this experiment, the emission noise variances and inducing input points are estimated, while all other parameters are fixed to the true values used in generating the data.

Fig. 2 shows the parameter estimation results. The estimates of the 39-dimensional noise variance of the emission functions are shown with the true noise variances, the true RBF kernel variances, and the sample variances of the synthetic data. The top row denotes the estimation results without the variance constraint, and the bottom row with the variance constraint. By comparing the two figures

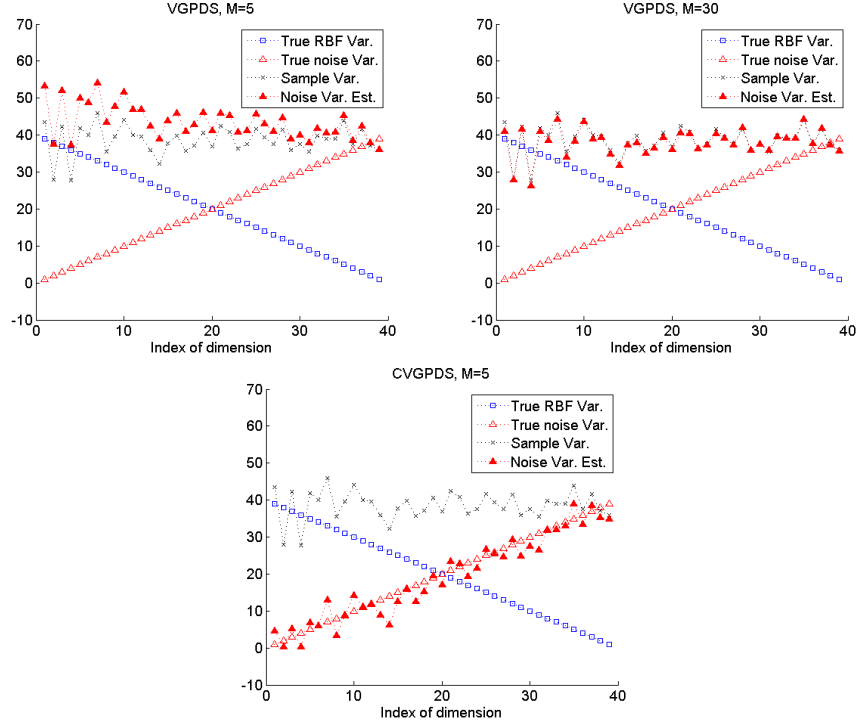

Figure 2: Results of parameter estimation: (top-left) VGPDS with M = 5, (top-right) VGPDS with M = 30, and (bottom) CVGPDS with M = 5

on the top row, we can confirm that the estimation result of the noise variance with $M = 30$ inducing input points is better than that with $M = 5$ inducing input points. This result is obvious in the sense that smaller values of $M$ produce more errors in the sparse approximation of the covariance matrix. However, both noise variance estimates are still different from the true values. By comparing the top and bottom rows, we can see that the proposed CVGPDS outperforms the VGPDS in terms of parameter estimation. Remarkably, the estimation result of the CVGPDS with $M = 5$ inducing input points is much better than the result of the VGPDS with $M = 30$. Based on these observations, we can conclude that the proposed CVGPDS is considerably more robust to the sparse approximation error compared to the VGPDS, as we claimed in Section 2.2.4.

### 3.2 Two-class classification using synthetic data

This section aims to show that when there is strong dependency over the observations, the proposed CVGPDS is a more appropriate model than the HMM for the classification task. To this end, we first generated several sets of two-class classification datasets with different degrees of dependency over the observations. The considered classification task is to map each input segment to one of two class labels. Using $s \in \{1, ..., S\}$ as the segment index, the synthetic dataset $\mathcal{D} = \{\mathbf{Y}_s, \mathbf{t}_s, l_s\}_{s=1}^S$ consists of $S$ segments, where the $s$-th segment has $N_s$ samples. Here, $\mathbf{Y}_s \in \mathbb{R}^{N_s \times D}$, $\mathbf{t}_s \in \mathbb{R}^{N_s}$, and $l_s$ are the observation data, time, and class label of the $s$-th segment, respectively. The synthetic dataset is generated as follows:

- Mean and kernel functions of two GPs $g_j(t)$ and $f_i(\mathbf{x})$ are defined as

$$
\begin{array}{llll}
g_j(t): & \mu_j^g(t) = a_j t + b_j, & k_j^g(t, t') = \mathbf{1}_{t=t'} \\
f_i(\mathbf{x}): & \mu_i^f(\mathbf{x}) = \sum_{z=1}^{Z_i} w_z N(\mathbf{x}; \mathbf{m}_i^z \mathbf{\Lambda}_i^z), & k_i^f(\mathbf{x}, \mathbf{x}') = \alpha_i \exp(-\omega_i ||\mathbf{x} - \mathbf{x}'||)
\end{array}
\tag{12}
$$

where $\{a_j, b_j\}$, $\{w_z, \mathbf{m}_i^z, \mathbf{\Lambda}_i^z\}$, and $\{\alpha_i, \omega_i\}$ are respectively the parameters of the linear, Gaussian mixture, and RBF kernel functions. The superscript $z$ denotes the component index of the Gaussian mixture, and $Z_i$ is the number of components in $f_i(\mathbf{x})$.

- For the $s$-th segment, $\{\mathbf{Y}_s, \mathbf{t}_s, l_s\}$,

  1. $l_s$ is selected as either class 1 or 2.
  2. $N_s$ is randomly selected from interval $[20, 30]$, and $\mathbf{t}_s$ is obtained by using Eq. (6).
  3. From $\mathbf{t}_s$, the mean vector $\boldsymbol{\mu}_j^g(\mathbf{t}_s)$ and covariance matrix $\mathbf{K}_j^g$ are obtained for $j = 1, ..., Q$. Let $\mathbf{X}_s \in \mathbb{R}^{N_s \times Q}$ be the latent state of the $s$-th segment. Then, the $j$-th column of $\mathbf{X}_s$ is generated by the $N_s$-dimensional Gaussian distribution $\mathcal{N}(\boldsymbol{\mu}_j^g(\mathbf{t}_s), \mathbf{K}_j^g)$.
  4. From $\mathbf{X}_s$, the mean vector $\boldsymbol{\mu}_i^f(\mathbf{X}_s)$ and covariance matrix $\mathbf{K}_i^f$ are obtained for $i = 1, ..., D$. Then, the $i$-th column of $\mathbf{Y}_s$ is generated by the $N_s$-dimensional Gaussian distribution, $\mathcal{N}(\boldsymbol{\mu}_i^f(\mathbf{X}_s), \mathbf{K}_i^f)$.

Note that parameter $\omega_i$ controls the degree of dependency over the observations. For instance, if $\omega_i$ decreases, the off-diagonal terms of the emission kernel matrix $\mathbf{K}_i^f$ increase, which means stronger correlations over the observations.

The experimental setups are as follows. The synthesized dataset consists of 200 segments in total (100 segments per class). The dimensions of the latent space and observation space are set to $Q = 2$ and $D = 5$, respectively. We use $6(= Z_i)$ components for the mean function of the emission kernel function. In this experiment, three datasets are synthesized and used to compare the CVGPDS and the HMM. When generating each dataset, we use two different $\omega_i$ values, one for each class, while all other parameters in Eq. (12) are shared between the two classes. As a result, the degree of correlation between the observations is the only factor that distinguishes the two classes. The three generated datasets have different degrees of correlation over the observations, as a result of setting different $\omega_i$ values for generating each dataset. In particular, the third dataset is constructed with two limitations of HMM such that it is well represented by an HMM. This could be achieved simply by changing the form of the mean function $\mu_j^g(t)$ from a linear to a step function, and setting $\omega_i = \infty$ so that each data sample is generated independently of the others. In the third dataset, the two classes are set to have different $\alpha_i$ values. The classification experiments are conducted using an HMM and CVGPDS.

Table 1: Classification accuracy for the two-class synthetic datasets (10-fold CV average [%]): All parameters except $\omega_i$ are set to be equal for classes 1 and 2. In the case of $\omega_i = \infty$, $\alpha_i$ are set to be different.

| $\omega_i$ (class 1 : class 2) | 0.1 : 0.5 | 1.0 : 2.0 | $\infty : \infty$ |
|---|---|---|---|
| HMM | 61.0 | 68.5 | 88.5 |
| CVGPDS | 78.0 | 79.0 | 92.0 |

Table 1 summarizes the classification performance of the HMM and CVGPDS for the three synthetic datasets. Remarkably, in all cases, the proposed CVGPDS outperforms the HMM, even in the case of $\omega_i = \infty$ (the fourth column), where we assumed the dataset follows HMM-like characteristics. Comparing the second and the third columns of Table 1, we can see that the performance of the HMM degrades by $6.5\%$ as $\omega_i$ becomes smaller, while the proposed CVGPDS almost maintains its performance with only $1.0\%$ reduction. This result demonstrates the superiority of the proposed CVGPDS in modeling data with strong correlations over the observations. Apparently, the HMM failed to distinguish the two classes with different degree of dependency over the observations. In contrast, the proposed CVGPDS distinguishes the two classes more effectively by capturing the different degrees of inter-dependencies over the observations incorporated in each class.

### 3.3 Phoneme classification

In this section, phoneme classification experiments is described on real speech data from the TIMIT database. The TIMIT database contains a total of 6300 phonetically rich utterances, each of which is manually segmented based on 61 phoneme transcriptions. Following the standard regrouping of phoneme labels [11], 61 phonemes are reduced to 48 phonemes selected for modeling. As observations, 39-dimensional Mel-frequency cepstral coefficients (MFCCs) (13 static coefficients, $\Delta$, and

$\Delta\Delta$) extracted from the speech signals with standard 25 ms frame size, and 10 ms frame shifts are used. The dimension of the latent space is set to $Q = 2$.

For the first phoneme classification experiment, 100 segments per phoneme are randomly selected using the phoneme boundary provided information in the TIMIT database. The number of inducing input points is set to $M = 10$. A 10-fold cross-validation test was conducted to evaluate the proposed model in comparison with an HMM that has three states and a single Gaussian distribution with a full covariance matrix per state. Parameters of the HMMs are estimated by using the conventional expectation-maximization (EM) algorithm with a maximum likelihood criterion.

Table 2: Classification accuracy on the 48-phoneme dataset (10-fold CV average [%]): 100 segments are used for training and testing each phoneme model

| HMM | VGPDS | CVGPDS |
|-----|-------|--------|
| 49.19 | 48.17 | 49.36 |

Table 2 shows the experimental results of a 48-phoneme classification. Compared to the HMM and VGPDS, the proposed CVGPDS performs more effectively.

For the second phoneme classification experiment, the TIMIT core test set consisting of 192 sentences is used for evaluation. We use the same 100 segments for training the phoneme models as in the first phoneme classification experiment. The size of the training dataset is smaller than that of conventional approaches due to our limited computational ability. When evaluating the models, we merge the labels of 48 phonemes into the commonly used 39 phonemes [11]. Given speech observations with boundary information, a sequence of log-likelihoods is obtained, and then a bigram is constructed to incorporate linguistic information into the classification score. In this experiment, the number of inducing input points is set to $M = 5$.

Table 3: Classification accuracy on the TIMIT core test set [%]: 100 segments are used for training each phoneme model

| HMM | VGPDS | CVGPDS |
|-----|-------|--------|
| 57.83 | 61.44 | 61.54 |

Table 3 shows the experimental results of phoneme classification for the TIMIT core test set. As shown by the results in Table 2, the proposed CVGPDS performed better than the HMM and VG-PDS. However, the classification accuracies in Table 3 are lower than the state-of-the-art phoneme classification results [12-13]. The reasons for low accuracy are as follows: 1) insufficient amount of data is used for training the model owing to limited availability of computational power; 2) a mixture model for the emission is not considered. These remaining issues need to be addressed for improved performance.

## 4    Conclusion

In this paper, a VGPDS-based acoustic model for phoneme classification was considered. The proposed acoustic model can represent the nonlinear latent dynamics and dependency among observations by GP priors. In addition, we introduced a variance constraint on the VGPDS. Although the proposed model could not achieve the state-of-the-art performance of phoneme classification, the experimental results showed that the proposed acoustic model has potential for speech modeling. For future works, extension to phonetic recognition and mixture of the VGPDS will be considered.

**Acknowledgments**

This work was supported by the National Research Foundation of Korea(NRF) grant funded by the Korea government(MEST) (No.2012-0005378 and No.2012-0000985)

# References

[1] F. Jelinek, "Continuous speech recognition by statistical methods," Proceedings of the IEEE, Vol.64, pp.532-556, 1976.

[2] M. Ostendorf, V. Digalakis, and J. Rohlicek, "From HMMs to segment models: A unified view of stochastic modeling for speech recognition," IEEE Trans. on Speech and Audio Processing, Vol.4, pp.360-378, 1996.

[3] L. Deng, D. Yu, and A. Acero, "Structured Speech Modeling," IEEE Trans. on Audio, Speech, and Language Processing, Vol.14, pp.1492-1504, 2006.

[4] C. E. Rasmussen and C. K. I. Williams, "Gaussian Process for Machine Learning," MIT Press, Cambridge, MA, 2006.

[5] N. D. Lawrence, "Probabilistic non-linear principal component analysis with Gaussian process latent variable models," Journal of Machine Learning Research (JMLR), Vol.6, pp.1783-1816, 2005.

[6] N. D. Lawrence, "Learning for larger datasets with the Gaussian process latent variable model," International Conference on Artificial Intelligence and Statistics (AISTATS), pp.243-250, 2007.

[7] M. K. Titsias and N. D. Lawrence, "Bayesian Gaussian Process Latent Variable Model," International Conference on Artificial Intelligence and Statistics (AISTATS), pp.844-851, 2010.

[8] J. Quiñonero-Candela and C. E. Rasmussen, "A Unifying View of Sparse Approximate Gaussian Process Regression," Journal of Machine Learning Research (JMLR), Vol.6, pp.1939-1959, 2005.

[9] A. C. Damianou, M. K. Titsias, and N. D. Lawrence, "Variational Gaussian Process Dynamical Systems," Advances in Neural Information Processing Systems (NIPS), 2011.

[10] J. M. Wang, D. J. Fleet, and A. Hertzmann, "Gaussian Process Dynamical Models for Human Motion," IEEE Trans. Pattern Analysis and Machine Intelligence, Vol.30, pp.283-298, 2008.

[11] K. F. Lee and H. W. Hon, "Speaker-independent phone recognition using hidden Markov models," IEEE Trans. on Acoustics, Speech and Signal Processing, vol.37, pp.1641-1648, 1989.

[12] A. Mohamed, G. Dahl, and G. Hinton, "Acoustic modeling using deep belief networks," IEEE Trans. on Audio, Speech, and Language Processing, Vol.20, no.1, pp. 14-22, 2012.

[13] F. Sha and L. K. Saul, "Large margin hidden markov models for automatic speech recognition," Advances in Neural Information Processing Systems (NIPS), 2007.

